# Modeling Dyadic Data with Binary Latent Factors

**Edward Meeds**
Department of Computer Science
University of Toronto
ewm@cs.toronto.edu

**Zoubin Ghahramani**
Department of Engineering
Cambridge University
zoubin@eng.cam.ac.uk

**Radford Neal**
Department of Computer Science
University of Toronto
radford@cs.toronto.edu

**Sam Roweis**
Department of Computer Science
University of Toronto
roweis@cs.toronto.edu

## Abstract

We introduce *binary matrix factorization*, a novel model for unsupervised matrix decomposition. The decomposition is learned by fitting a non-parametric Bayesian probabilistic model with binary latent variables to a matrix of dyadic data. Unlike bi-clustering models, which assign each row or column to a single cluster based on a categorical hidden feature, our binary feature model reflects the prior belief that items and attributes can be associated with more than one latent cluster at a time. We provide simple learning and inference rules for this new model and show how to extend it to an infinite model in which the number of features is not a priori fixed but is allowed to grow with the size of the data.

## 1 Distributed representations for dyadic data

One of the major goals of probabilistic unsupervised learning is to discover underlying or hidden structure in a dataset by using latent variables to describe a complex data generation process. In this paper we focus on *dyadic data*: our domains have two finite sets of objects/entities and observations are made on *dyads* (pairs with one element from each set). Examples include sparse matrices of movie-viewer ratings, word-document counts or product-customer purchases. A simple way to capture structure in this kind of data is to do "bi-clustering" (possibly using mixture models) by grouping the rows and (independently or simultaneously) the columns[6, 13, 9]. The modelling assumption in such a case is that movies come in $K$ types and viewers in $L$ types and that knowing the type of movie and type of viewer is sufficient to predict the response. Clustering or mixture models are quite restrictive – their major disadvantage is that they do not admit a componential or distributed representation because items cannot simultaneously belong to several classes. (A movie, for example, might be explained as coming from a cluster of "dramas" or "comedies"; a viewer as a "single male" or as a "young mother".) We might instead prefer a model (e.g. [10, 5]) in which objects can be assigned to multiple latent clusters: a movie might be a drama and have won an Oscar and have subtitles; a viewer might be single and female and a university graduate. Inference in such models falls under the broad area of *factorial learning* (e.g. [7, 1, 3, 12]), in which multiple interacting latent causes explain each observed datum.

In this paper, we assume that both data items (rows) and attributes (columns) have this kind of componential structure: each item (row) has associated with it an unobserved vector of $K$ binary features; similarly each attribute (column) has a hidden vector of $L$ binary features. Knowing the features of the item and the features of the attribute are sufficient to generate (before noise) the response at that location in the matrix. In effect, we are factorizing a real-valued data (response) matrix $\mathbf{X}$ into (a distribution defined by) the product $\mathbf{UWV}^\top$, where $\mathbf{U}$ and $\mathbf{V}$ are binary feature matrices, and $\mathbf{W}$ is a real-valued weight matrix. Below, we develop this *binary matrix factorization*

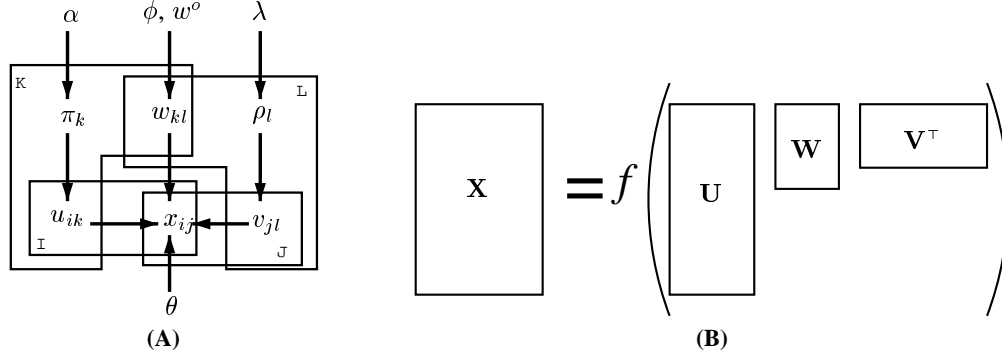

Figure 1: **(A)** The graphical model representation of the linear-Gaussian BMF model. The concentration parameter and Beta weights for the columns of $\mathbf{X}$ are represented by the symbols $\lambda$ and $\rho_l$. **(B)** BMF shown pictorally.

(BMF) model using Bayesian non-parametric priors over the number and values of the unobserved binary features and the unknown weights.

## 2    BMF model description

Binary matrix factorization is a model of an $I \times J$ dyadic data matrix $\mathbf{X}$ with exchangeable rows and columns. The entries of $\mathbf{X}$ can be real-valued, binary, or categorial; BMF models suitable for each type are described below. Associated with each row is a latent binary feature vector $\mathbf{u}_i$; similarly each column has an unobserved binary vector $\mathbf{v}_j$. The primary parameters are represented by a matrix $\mathbf{W}$ of interaction weights. $\mathbf{X}$ is generated by a fixed observation process $f(\cdot)$ applied (elementwise) to the linear inner product of the features and weights, which is the "factorization" or approximation of the data:

$$\mathbf{X} \mid \mathbf{U}, \mathbf{V}, \mathbf{W} \ \sim \ f(\mathbf{U}\mathbf{W}\mathbf{V}^\top, \Theta) \tag{1}$$

where $\Theta$ are extra parameters specific to the model variant. Three possible parametric forms for the noise (observation) distribution $f$ are: Gaussian, with mean $\mathbf{U}\mathbf{W}\mathbf{V}^\top$ and covariance $(1/\theta)\,\mathbf{I}$; logistic, with mean $1/\left(1 + \exp(-\mathbf{U}\mathbf{W}\mathbf{V}^\top)\right)$; and Poisson, with mean (and variance) $\mathbf{U}\mathbf{W}\mathbf{V}^\top$. Other parametric forms are also possible. For illustrative purposes, we will use the linear-Gaussian model throughout this paper; this can be thought of as a two-sided version of the linear-Gaussian model found in [5].

To complete the description of the model, we need to specify prior distributions over the feature matrices $\mathbf{U}, \mathbf{V}$ and the weights $\mathbf{W}$. We adopt the same priors over binary matrices as previously described in [5]. For finite sized matrices $\mathbf{U}$ with $I$ rows and $K$ columns, we generate a bias $\pi_k$ independently for each column $k$ using a Beta prior (denoted $\mathcal{B}$) and then conditioned on this bias generate the entries in column $k$ independently from a Bernoulli with mean $\pi_k$.

$$\pi_k \mid \alpha, K \ \sim \ \mathcal{B}\left(\alpha/K, \ \beta\right) \qquad\qquad \alpha \mid a_\alpha, b_\alpha \ \sim \ \mathcal{G}\left(a_\alpha, \ b_\alpha\right)$$

$$\mathbf{U} \mid \boldsymbol{\pi} \ \sim \ \prod_{i=1}^{I}\prod_{k=1}^{K} \pi_k^{u_{ik}}\left(1 - \pi_k\right)^{1-u_{ik}} \ = \ \prod_{k=1}^{K} \pi_k^{n_k}\left(1 - \pi_k\right)^{I-n_k}$$

where $n_k = \sum_i u_{ik}$. The hyperprior on the concentration $\alpha$ is a Gamma distribution (denoted $\mathcal{G}$), whose shape and scale hyperparameters control the expected fraction of zeros/ones in the matrix. The biases $\boldsymbol{\pi}$ are easily integrated out, which creates dependencies between the rows, although they remain exchangeable. The resulting prior depends only on the number $n_k$ of active features in each column. An identical prior is used on $\mathbf{V}$, with $J$ rows and $L$ columns, but with different concentration prior $\lambda$. The variable $\beta$ was set to 1 for all experiments.

The appropriate prior distribution over weights depends on the observation distribution $f(\cdot)$. For the linear-Gaussian variant, a convenient prior on $\mathbf{W}$ is a matrix normal with prior mean $\mathbf{W}^o$ and

covariance $(1/\phi)\,\mathbf{I}$. The scale $\phi$ of the weights and output precision $\theta$ (if needed) have Gamma hyperpriors:

$$
\begin{aligned}
\mathbf{W} \mid \mathbf{W}^o, \phi &\sim \mathcal{N}\left(\mathbf{W}^o,\ (1/\phi)\,\mathbf{I}\right) \\
\phi \mid a_\phi, b_\phi &\sim \mathcal{G}\left(a_\phi,\ b_\phi\right) \\
\theta \mid a_\theta, b_\theta &\sim \mathcal{G}\left(a_\theta,\ b_\theta\right)
\end{aligned}
$$

In certain cases, when the prior on the weights is conjugate to the output distribution model $f$, the weights may be analytically integrated out, expressing the marginal distribution of the data $\mathbf{X}|\mathbf{U}, \mathbf{V}$ only in terms of the binary features. This is true, for example, when we place a Gaussian prior on the weights and use a linear-Gaussian output process.

Remarkably, the Beta-Bernoulli prior distribution over $\mathbf{U}$ (and similarly $\mathbf{V}$) can easily be extended to the case where $K \to \infty$, creating a distribution over binary matrices with a fixed number $I$ of exchangeable rows and a potentially infinite number of columns (although the expected number of columns which are not entirely zero remains finite). Such a distribution, the Indian Buffet Process (IBP) was described by [5] and is analogous to the Dirichlet process and the associated Chinese restaurant process (CRP) [11]. Fortunately, as we will see, inference with this infinite prior is not only tractable, but is also nearly as efficient as the finite version.

## 3 Inference of features and parameters

As with many other complex hierarchical Bayesian models, exact inference of the latent variables $\mathbf{U}$ and $\mathbf{V}$ in the BMF model is intractable (ie there is no efficient way to sample exactly from the posterior nor to compute its exact marginals). However, as with many other non-parametric Bayesian models, we can employ Markov Chain Monte Carlo (MCMC) methods to create an iterative procedure which, if run for sufficiently long, will produce correct posterior samples.

### 3.1 Finite binary latent feature matrices

The posterior distribution of a single entry in $\mathbf{U}$ (or $\mathbf{V}$) given all other model parameters is proportional to the product of the conditional prior and the data likelihood. The conditional prior comes from integrating out the biases $\pi$ in the Beta-Bernoulli model and is proportional the number of active entries in other rows of the same column plus a term for new activations. Gibbs sampling for single entries of $\mathbf{U}$ (or $\mathbf{V}$) can be done using the following updates:

$$
\begin{aligned}
P\left(u_{ik}=1|\mathbf{U}_{-ik}, \mathbf{V}, \mathbf{W}, \mathbf{X}\right) &= C\left(\alpha/K + n_{-i,k}\right) P\left(\mathbf{X}|\mathbf{U}_{-ik}, u_{ik}=1, \mathbf{V}, \mathbf{W}\right) \quad (2) \\
P\left(u_{ik}=0|\mathbf{U}_{-ik}, \mathbf{V}, \mathbf{W}, \mathbf{X}\right) &= C\left(\beta + (I-1) - n_{-i,k}\right) P\left(\mathbf{X}|\mathbf{U}_{-ik}, u_{ik}=0, \mathbf{V}, \mathbf{W}\right) \quad (3)
\end{aligned}
$$

where $n_{-i,k} = \sum_{h\neq i} u_{hk}$, $\mathbf{U}_{-ik}$ excludes entry $ik$, and $C$ is a normalizing constant. (Conditioning on $\alpha, K$ and $\theta$ is implicit.) When conditioning on $\mathbf{W}$, we only need to calculate the ratio of likelihoods corresponding to row $i$. (Note that this is not the case when the weights are integrated out.) This ratio is a simple function of the model's predictions $\hat{x}_{ij}^+ = \sum_{hl} u_{ih} v_{jl} w_{hl}$ (when $u_{ik}=1$) and $\hat{x}_{ij}^- = \sum_{hl} u_{ih} v_{jl} w_{hl}$ (when $u_{ik}=0$). In the linear-Gaussian case:

$$
\log \frac{P\left(u_{ik}=1|\mathbf{U}_{-ik}, \mathbf{V}, \mathbf{W}, \mathbf{X}\right)}{P\left(u_{ik}=0|\mathbf{U}_{-ik}, \mathbf{V}, \mathbf{W}, \mathbf{X}\right)} = \log \frac{\left(\alpha/K + n_{-i,k}\right)}{\left(\beta + (I-1) - n_{-i,k}\right)} - \frac{1}{2}\sum_j \theta_{ij}\left[(x_{ij} - \hat{x}_{ij}^+)^2 - (x_{ij} - \hat{x}_{ij}^-)^2\right]
$$

In the linear-Gaussian case, we can easily derive analogous Gibbs sampling updates for the weights $\mathbf{W}$ and hyperparameters. To simplify the presentation, we consider a "vectorized" representation of our variables. Let $\mathbf{x}$ be an $IJ$ column vector taken column-wise from $\mathbf{X}$, $\mathbf{w}$ be a $KL$ column vector taken column-wise from $\mathbf{W}$ and $\mathbf{A}$ be a $IJ \times KL$ binary matrix which is the kronecker product $\mathbf{V} \otimes \mathbf{U}$. (In "Matlab notation", $\mathbf{x} = \mathbf{X}(:)$, $\mathbf{w} = \mathbf{W}(:)$ and $\mathbf{A} = \texttt{kron}(\mathbf{V}, \mathbf{U})$.) In this notation, the data distribution is written as: $\mathbf{x}|\mathbf{A}, \mathbf{w}, \theta \sim \mathcal{N}\left(\mathbf{A}\mathbf{w},\ (1/\theta)\,\mathbf{I}\right)$. Given values for $\mathbf{U}$ and $\mathbf{V}$, samples can be drawn for $\mathbf{w}$, $\phi$, and $\theta$ using the following posterior distributions (where conditioning on $\mathbf{w}^o, \phi, \theta, a_\phi, b_\phi, a_\theta, b_\theta$ is implicit):

$$
\mathbf{w} \mid \mathbf{x}, \mathbf{A} \sim \mathcal{N}\left(\left(\theta\mathbf{A}^\top\mathbf{A} + \phi\mathbf{I}\right)^{-1}\left(\theta\mathbf{A}^\top\mathbf{x} + \phi\mathbf{w}^o\right),\ \left(\theta\mathbf{A}^\top\mathbf{A} + \phi\mathbf{I}\right)^{-1}\right)
$$

$$\phi \mid \mathbf{w} \quad \sim \quad \mathcal{G}\left(a_\phi + KL/2, \ \left(b_\phi + \frac{1}{2}\left(\mathbf{w} - \mathbf{w}^o\right)^\top \left(\mathbf{w} - \mathbf{w}^o\right)\right)\right)$$

$$\theta \mid \mathbf{x}, \mathbf{A}, \mathbf{w} \quad \sim \quad \mathcal{G}\left(a_\theta + IJ/2, \ \left(b_\theta + \frac{1}{2}\left(\mathbf{x} - \mathbf{A}\mathbf{w}\right)^\top \left(\mathbf{x} - \mathbf{A}\mathbf{w}\right)\right)\right)$$

Note that we do not have to explicitly compute the matrix $\mathbf{A}$. For computing the posterior of linear-Gaussian weights, the matrix $\mathbf{A}^\top \mathbf{A}$ can be computed as $\mathbf{A}^\top \mathbf{A} = \texttt{kron}(\mathbf{V}^\top \mathbf{V}, \mathbf{U}^\top \mathbf{U})$. Similarly, the expression $\mathbf{A}^\top \mathbf{x}$ is constructed by computing $\mathbf{U}^\top \mathbf{X} \mathbf{V}$ and taking the elements column-wise.

### 3.2 Infinite binary latent feature matrices

One of the most elegant aspects of non-parametric Bayesian modeling is the ability to use a prior which allows a countably infinite number of latent features. The number of instantiated features is automatically adjusted during inference and depends on the amount of data and how many features it supports. Remarkably, we can do MCMC sampling using such infinite priors with essentially no computational penalty over the finite case. To derive these updates (e.g. for row $i$ of the matrix $\mathbf{U}$), it is useful to consider partitioning the columns of $\mathbf{U}$ into two sets as shown below.

Let set A have at least one non-zero entry in rows other than $i$. Let set B be all other columns, including the set of columns where the only non-zero entries are found in row $i$ and the countably infinite number of all-zero columns. Sampling values for elements in row $i$ of set A given everything else is straightforward, and involves Gibbs updates almost identical to those in the finite case handled by equations (2) and (3); as $K \to \infty$ and $k$ in set A we get:

| set A | | | | | set B | | | | | |
|---|---|---|---|---|---|---|---|---|---|---|
| 0 | 1 | 0 | 0 | 1 | 0 | 0 | 0 | 0 | 0 | $\cdots$ |
| 0 | 0 | 1 | 0 | 0 | 0 | 0 | 0 | 0 | 0 | $\cdots$ |
| 1 | 1 | 0 | 0 | 1 | 0 | 0 | 0 | 0 | 0 | $\cdots$ |
| 1 | 0 | 0 | 1 | 1 | 0 | 0 | 0 | 0 | 0 | $\cdots$ |
| 1 | 1 | 0 | 0 | 1 | 0 | 1 | 0 | 1 | 0 | row $i$ |
| 0 | 1 | 0 | 0 | 0 | 0 | 0 | 0 | 0 | 0 | $\cdots$ |
| 0 | 0 | 0 | 1 | 0 | 0 | 0 | 0 | 0 | 0 | $\cdots$ |
| 1 | 0 | 0 | 0 | 1 | 0 | 0 | 0 | 0 | 0 | $\cdots$ |

$$P\left(u_{ik} = 1 | \mathbf{U}_{-ik}, \mathbf{V}, \mathbf{W}\right) \quad = \quad C \cdot n_{-i,k} \, P\left(\mathbf{X} | \mathbf{U}_{-ik}, u_{ik} = 1, \mathbf{V}, \mathbf{W}\right) \tag{4}$$

$$P\left(u_{ik} = 0 | \mathbf{U}_{-ik}, \mathbf{V}, \mathbf{W}\right) \quad = \quad C \cdot (\beta + I - 1 - n_{-i,k}) \, P\left(\mathbf{X} | \mathbf{U}_{-ik}, u_{ik} = 0, \mathbf{V}, \mathbf{W}\right) \tag{5}$$

When sampling new values for set B, the columns are exchangeable, and so we are really only interested in the number of entries $n_B^\star$ in set B which will be turned on in row $i$. Sampling the number of entries set to 1 can be done with Metropolis-Hastings updates. Let $J\left(n_B^\star | n_B\right) = \text{Poisson}\left(n_B^\star | \alpha / (\beta + I - 1)\right)$ be the proposal distribution for a move which replaces the current $n_B$ active entries with $n_B^\star$ active entries in set B. The reverse proposal is $J\left(n_B | n_B^\star\right)$. The acceptance probability is $\min\left(1, r_{n_B \to n_B^\star}\right)$, where $r_{n_B \to n_B^\star}$ is

$$\frac{P\left(n_B^\star | \mathbf{X}\right) J\left(n_B | n_B^\star\right)}{P\left(n_B | \mathbf{X}\right) J\left(n_B^\star | n_B\right)} = \frac{P\left(\mathbf{X} | n_B^\star\right) \text{Poisson}(n_B^\star | \alpha / (\beta + I - 1)) J\left(n_B | n_B^\star\right)}{P\left(\mathbf{X} | n_B\right) \text{Poisson}(n_B | \alpha / (\beta + I - 1)) J\left(n_B^\star | n_B\right)} = \frac{P\left(\mathbf{X} | n_B^\star\right)}{P\left(\mathbf{X} | n_B\right)} \tag{6}$$

This assumes a conjugate situation in which the weights $\mathbf{W}$ are explicitly integrated out of the model to compute the marginal likelihood $P(\mathbf{X} | n_B^\star)$. In the non-conjugate case, a more complicated proposal is required. Instead of proposing $n_B^\star$, we jointly propose $n_B^\star$ and associated feature parameters $\mathbf{w}_B^\star$ from their prior distributions. In the linear-Gaussian model, where $\mathbf{w}_B^\star$ is a set of weights for features in set B, the proposal distribution is:

$$J\left(n_B^\star, \mathbf{w}_B^\star | n_B, \mathbf{w}_B\right) = \text{Poisson}\left(n_B^\star | \alpha / (\beta + I - 1)\right) \text{Normal}\left(\mathbf{w}_B^\star | n_B^\star, \phi\right) \tag{7}$$

We need actually sample only the finite portion of $\mathbf{w}_B^\star$ where $u_{ik} = 1$. As in the conjugate case, the acceptance ratio reduces to the ratio of data likelihoods:

$$r_{n_B, \mathbf{w}_B \to n_B^\star, \mathbf{w}_B^\star} \quad = \quad \frac{P\left(\mathbf{X} | n_B^\star, \mathbf{w}_B^\star\right)}{P\left(\mathbf{X} | n_B, \mathbf{w}_B\right)} \tag{8}$$

### 3.3 Faster mixing transition proposals

The Gibbs updates described above for the entries of $\mathbf{U}, \mathbf{V}$ and $\mathbf{W}$ are the simplest moves we could make in a Markov Chain Monte Carlo inference procedure for the BMF model. However, these

limited local updates may result in extremely slow mixing. In practice, we often implement larger moves in indicator space using, for example, Metropolis-Hastings proposals on multiple features for row $i$ simultaneously. For example, we can propose new values for several columns in row $i$ of matrix $\mathbf{U}$ by sampling feature values independently from their conditional priors. To compute the reverse proposal, we imagine forgetting the current configuration of those features for row $i$ and compute the probability under the conditional prior of proposing the current configuration. The acceptance probability of such a proposal is (the maximum of unity and) the ratio of likelihoods between the new proposed configuration and the current configuration.

Split-merge moves may also be useful for efficiently sampling from the posterior distribution of the binary feature matrices. Jain and Neal [8] describe split-merge algorithms for Dirichlet process mixture models with non-conjugate component distributions. We have developed and implemented similar split-merge proposals for binary matrices with IBP priors. Due to space limitations, we present here only a sketch of the procedure. Two nonzero entries in $\mathbf{U}$ are selected uniformly at random. If they are in the same column, we propose splitting that column; if they are in different columns, we propose merging their columns. The key difference between this algorithm and the Jain and Neal algorithm is that the binary features are not constrained to sum to unity in each row. Our split-merge algorithm also performs restricted Gibbs scans on columns of $\mathbf{U}$ to increase acceptance probability.

### 3.4 Predictions

A major reason for building generative models of data is to be able to impute missing data values given some observations. In the linear-Gaussian model, the predictive distribution at each iteration of the Markov chain is a Gaussian distribution. The interaction weights can be analytically integrated out at each iteration, also resulting in a Gaussian posterior, removing sampling noise contributed by having the weights explicitly represented. Computing the exact predictive distribution, however, conditional only on the model hyperparameters, is analytically intractable: it requires integrating over all binary matrices $\mathbf{U}$ and $\mathbf{V}$, and all other *nuisance* parameters (e.g., the weights and precisions). Instead we integrate over these parameters implicitly by averaging predictive distributions from many MCMC iterations. This posterior, which is conditional only on the observed data and hyperparameters, is highly complex, potentially multimodal, and non-linear function of the observed variables.

By averaging predictive distributions, our algorithm implicitly integrates over $\mathbf{U}$ and $\mathbf{V}$. In our experiments, we show samples from the posteriors of $\mathbf{U}$ and $\mathbf{V}$ to help explain what the model is doing, but we stress that the posterior may have significant mass on many possible binary matrices. The number of features and their degrees of overlap will vary over MCMC iterations. Such variation will depend, for example, on the current value of $\alpha$ and $\lambda$ (higher values will result in more features) and precision values (higher weight precision results in less variation in weights).

## 4 Experiments

### 4.1 Modified "bars" problem

A toy problem commonly used to illustrate additive feature or multiple cause models is the *bars problem* ([2, 12, 1]). Vertical and horizontal bars are combined in some way to generate data samples. The goal of the illustration is to show recovery of the latent structure in the form of bars. We have modified the typical usage of bars to accommodate the linear-Gaussian BMF with infinite features. Data consists of $I$ vectors of size $8^2$ where each vector can be reshaped into a square image. The generation process is as follows: since $\mathbf{V}$ has the same number of rows as the dimension of the images, $\mathbf{V}$ is fixed to be a set of vertical and horizontal bars (when reshaped into an image). $\mathbf{U}$ is sampled from the IBP, and global precisions $\theta$ and $\phi$ are set to $1/2$. The weights $\mathbf{W}$ are sampled from zero mean Gaussians. Model estimates of $\mathbf{U}$ and $\mathbf{V}$ were initialized from an IBP prior.

In Figure 2 we demonstrate the performance of the linear-Gaussian BMF on the bars data. We train the BMF with 200 training examples of the type shown in the top row in Figure 2. Some examples have their bottom halves labeled *missing* and are shown in the Figure with constant grey values. To handle this, we resample their values at each iteration of the Markov chain. The bottom row shows the expected reconstruction using MCMC samples of $\mathbf{U}$, $\mathbf{V}$, and $\mathbf{W}$. Despite the relatively high

noise levels in the data, the model is able to capture the complex relationships between bars and weights. The reconstruction of vertical bars is very good. The reconstruction of horizontal bars is good as well, considering that the model has no information regarding the existence of horizontal bars on the bottom half.

**(A)** Data samples

**(B)** Noise-free data

**(C)** Initial reconstruction

**(D)** Mean reconstruction

**(E)** Nearest neighbour

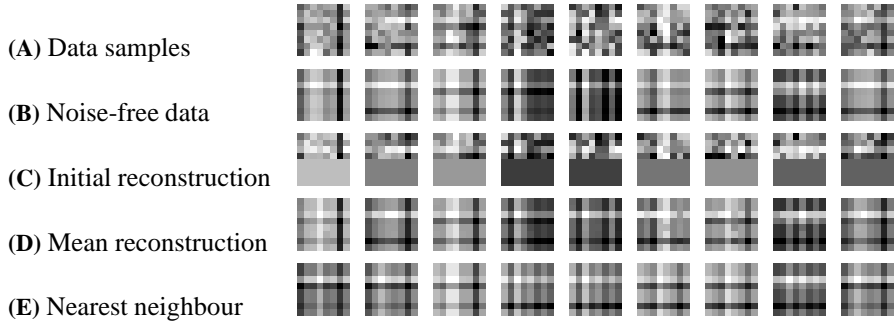

Figure 2: Bars reconstruction. **(A)** Bars randomly sampled from the complete dataset. The bottom half of these bars were removed and labeled *missing* during learning. **(B)** Noise-free versions of the same data. **(C)** The initial reconstruction. The missing values have been set to their expected value, $0$, to highlight the missing region. **(D)** The average MCMC reconstruction of the entire image. (E) Based solely on the information in the top-half of the original data, these are the noise-free nearest neighbours in pixel space.

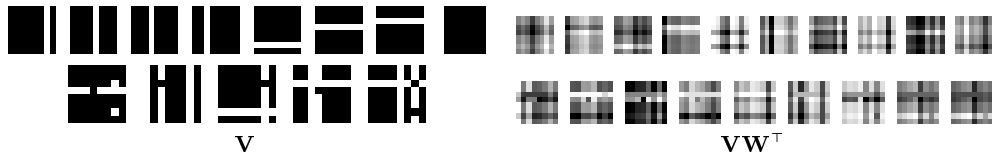

Figure 3: Bars features. The top row shows values of $\mathbf{V}$ and $\mathbf{WV}^\top$ used to generate the data. The second row shows a sample of $\mathbf{V}$ and $\mathbf{WV}^\top$ from the Markov chain. $\mathbf{WV}^\top$ can be thought of as a set of basis images which can be added together with binary coefficients ($\mathbf{U}$) to create images.

By examining the features captured by the model, we can understand the performance just described. In Figure 3 we show the generating, or *true*, values of $\mathbf{V}$ and $\mathbf{WV}^\top$ along with one sample of those features from the Markov chain. Because the model is generated by adding multiple $\mathbf{WV}^\top$ basis images shown on the right of Figure 3, multiple bars are used in each image. This is reflected in the captured features. The learned $\mathbf{WV}^\top$ are fairly similar to the generating $\mathbf{WV}^\top$, but the former are composed of overlapping bar structure (learned $\mathbf{V}$).

## 4.2 Digits

In Section 2 we briefly stated that BMF can be applied to data models other than the linear-Gaussian model. We demonstrate this with a *logistic* BMF applied to binarized images of handwritten digits. We train logistic BMF with 100 examples each of digits 1, 2, and 3 from the USPS dataset. In the first five rows of Figure 4 we again illustrate the ability of BMF to impute missing data values. The top row shows all 16 samples from the dataset which had their bottom halves labeled *missing*. Missing values are filled-in at each iteration of the Markov chain. In the third and fourth rows we show the mean and mode ($P(x_{ij} = 1) > 0.5$) of the BMF reconstruction. In the bottom row we have shown the nearest neighbors, in pixel space, to the training examples based only on the top halves of the original digits.

In the last three rows of Figure 4 we show the features captured by the model. In row F, we show the average image of the data which have each feature in $\mathbf{U}$ on. It is clear that some row features have distinct digit forms and others are overlapping. In row G, the basis images $\mathbf{WV}^\top$ are shown. By adjusting the features that are non-zero in each row of $\mathbf{U}$, images are composed by adding basis images together. Finally, in row H we show $\mathbf{V}$. These pixel features mask out different regions in

pixel space, which are weighted together to create the basis images. Note that there are $K$ features in rows F and G, and $L$ features in row H.

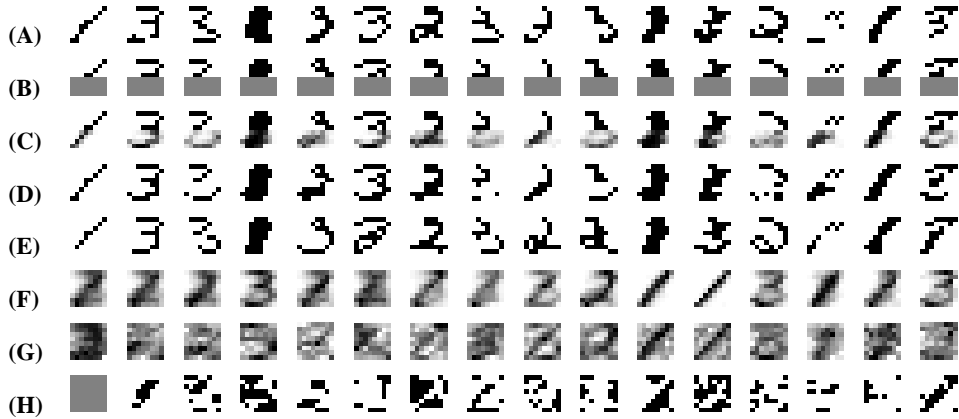

Figure 4: Digits reconstruction. **(A)** Digits randomly sampled from the complete dataset. The bottom half of these digits were removed and labeled *missing* during learning. **(B)** The data shown to the algorithm. The top half is the original data value. **(C)** The mean of the reconstruction for the bottom halves. **(D)** The mode reconstruction of the bottom halves. **(E)** The nearest neighbours of the original data are shown in the bottom half, and were found based solely on the information from the top halves of the images. **(F)** The average of all digits for each $\mathbf{U}$ feature. **(G)** The feature $\mathbf{W}\mathbf{V}^\top$ reshaped in the form of digits. By adding these features together, which the $\mathbf{U}$ features do, reconstructions of the digits is possible. **(H)** $\mathbf{V}$ reshaped into the form of digits. The first image represents a bias feature.

## 4.3 Gene expression data

Gene expression data is able to exhibit multiple and overlapping clusters simultaneously; finding models for such complex data is an interesting and active research area ([10], [13]). The plaid model[10], originally introduced for analysis of gene expression data, can be thought of as a non-Bayesian special case of our model in which the matrix $\mathbf{W}$ is diagonal and the number of binary features is fixed. Our goal in this experiment is merely to illustrate qualitatively the ability of BMF to find multiple clusters in gene expression data, some of which are overlapping, others non-overlapping. The data in this experiment consists of rows corresponding to genes and columns corresponding to patients; the patients suffer from one of two types of acute Leukemia [4]. In Figure 5 we show the factorization produced by the final state in the Markov chain. The rows and columns of the data and its expected reconstruction are ordered such that contiguous regions in $\mathbf{X}$ were observable. Some of the many feature pairings are highlighted. The BMF clusters consist of broad, overlapping clusters, and small, non-overlapping clusters. One of the interesting possibilities of using BMF to model gene expression data would be to fix certain columns of $\mathbf{U}$ or $\mathbf{V}$ with knowledge gained from experiments or literature, and to allow the model to add new features that help explain the data in more detail.

## 5 Conclusion

We have introduced a new model, *binary matrix factorization*, for unsupervised decomposition of dyadic data matrices. BMF makes use of non-parametric Bayesian methods to simultaneously discover binary distributed representations of both rows and columns of dyadic data. The model explains each row and column entity using a componential code composed of multiple binary latent features along with a set of parameters describing how the features interact to create the observed responses at each position in the matrix. BMF is based on a hierarchical Bayesian model and can be naturally extended to make use of a prior distribution which permits an infinite number of features, at very little extra computational cost. We have given MCMC algorithms for posterior inference of both the binary factors and the interaction parameters conditioned on some observed data, and

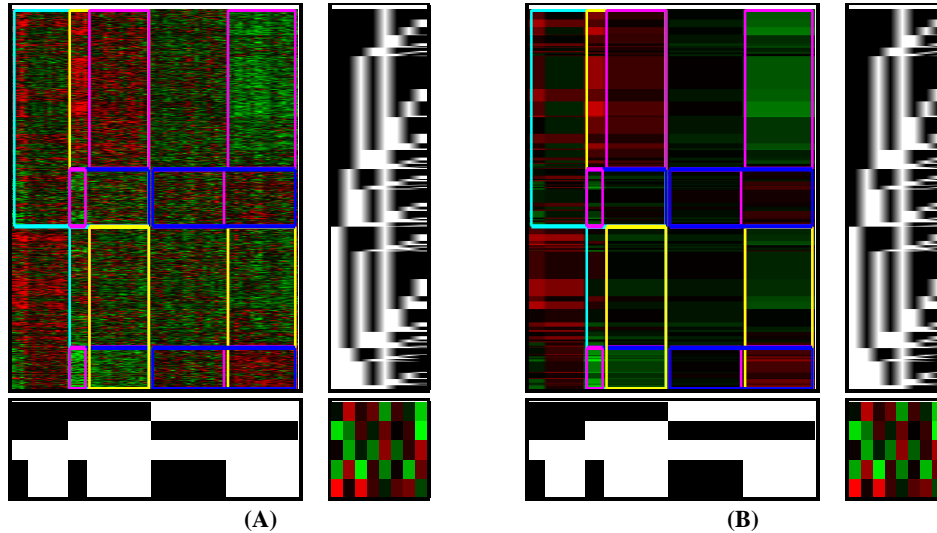

Figure 5: Gene expression results. **(A)** The top-left is $\mathbf{X}$ sorted according to contiguous features in the final $\mathbf{U}$ and $\mathbf{V}$ in the Markov chain. The bottom-left is $\mathbf{V}^\top$ and the top-right is $\mathbf{U}$. The bottom-right is $\mathbf{W}$. **(B)** The same as **(A)**, but the expected value of $\mathbf{X}$, $\hat{\mathbf{X}} = \mathbf{UWV}^\top$. We have highlighted regions that have both $u_{ik}$ and $v_{jl}$ on. For clarity, we have only shown the (at most) two largest contiguous regions for each feature pair.

demonstrated the model's ability to capture overlapping structure and model complex joint distributions on a variety of data. BMF is fundamentally different from bi-clustering algorithms because of its distributed latent representation and from factorial models with continuous latent variables which interact linearly to produce the observations. This allows a much richer latent structure, which we believe makes BMF useful for many applications beyond the ones we outlined in this paper.

## References

[1] P. Dayan and R. S. Zemel. Competition and multiple cause models. *Neural Computation*, 7(3), 1995.

[2] P. Foldiak. Forming sparse representations by local anti-Hebbian learning. *Biological Cybernetics*, 64, 1990.

[3] Z. Ghahramani. Factorial learning and the EM algorithm. In *NIPS*, volume 7. MIT Press, 1995.

[4] T. R. Golub, D. K. Slonim, P. Tamayo, C. Huard, M. Gaasenbeek, J. P. Mesirov, H. Coller, M. L. Loh, J. R. Downing, M. A. Caligiuri, C. D. Bloomfield, and E. S. Lander. Molecular classification of cancer: Class discovery and class prediction by gene expression monitoring. *Science*, 286(5439), 1999.

[5] T. Griffiths and Z. Ghahramani. Infinite latent feature models and the Indian buffet process. In *NIPS*, volume 18. MIT Press, 2005.

[6] J. A. Hartigan. Direct clustering of a data matrix. *Journal of the American Statistical Association*, 67, 1972.

[7] G. Hinton and R. S. Zemel. Autoencoders, minimum description length, and Helmholtz free energy. In *NIPS*, volume 6. Morgan Kaufmann, 1994.

[8] S. Jain and R. M. Neal. Splitting and merging for a nonconjugate Dirichlet process mixture model. To appear in *Bayesian Analysis*.

[9] C. Kemp, J. B. Tenebaum, T. L. Griffiths, T. Yamada, and N. Ueda. Learning systems of concepts with an infinite relational model. *Proceedings of the Twenty-First National Conference on Artificial Intelligence*, 2006.

[10] L. Lazzeroni and A. Owen. Plaid models for gene expression data. *Statistica Sinica*, 12, 2002.

[11] J. Pitman. Combinatorial stochastic processes. Lecture Notes for St. Flour Course, 2002.

[12] E. Saund. A multiple cause mixture model for unsupervised learning. *Neural Computation*, 7(1), 1994.

[13] R. Tibshirani, T. Hastie, M. Eisen, D. Ross, D. Botstein, and P. Brown. Clustering methods for the analysis of DNA microarray data. Technical report, Stanford University, 1999. Department of Statistics.
